# MAP Estimation for Graphical Models by Likelihood Maximization

**Akshat Kumar**
Department of Computer Science
University of Massachusetts
Amherst, MA
akshat@cs.umass.edu

**Shlomo Zilberstein**
Department of Computer Science
University of Massachusetts
Amherst, MA
shlomo@cs.umass.edu

## Abstract

Computing a *maximum a posteriori* (MAP) assignment in graphical models is a crucial inference problem for many practical applications. Several provably convergent approaches have been successfully developed using linear programming (LP) relaxation of the MAP problem. We present an alternative approach, which transforms the MAP problem into that of inference in a mixture of simple Bayes nets. We then derive the Expectation Maximization (EM) algorithm for this mixture that also monotonically increases a lower bound on the MAP assignment until convergence. The update equations for the EM algorithm are remarkably simple, both conceptually and computationally, and can be implemented using a graph-based message passing paradigm similar to max-product computation. Experiments on the real-world protein design dataset show that EM's convergence rate is significantly higher than the previous LP relaxation based approach MPLP. EM also achieves a solution quality within $95\%$ of optimal for most instances.

## 1  Introduction

Graphical models provide an effective framework to model complex systems via simpler local interactions and also provide an insight into the structure of the underlying probabilistic model. In particular, we focus on the class of undirected models called *Markov random fields* (MRFs) for which the joint distribution can be specified as the product of potential functions over the cliques of the graph. For many practical problems modeled using MRFs, finding the maximum a posteriori (MAP) assignment or the most probable assignment to the variables in the graph is a key inference problem. For example, MAP estimation has been applied to image processing in computer vision [17, 11], protein design and protein side-chain prediction problems [17, 11], and natural language processing [7]. Finding the MAP assignment is NP-hard in general except for tree-structured graphs and graphs with bounded treewidth [5, 3]. This further underscores the need for developing scalable approximation algorithms that provide good solution quality.

Recently, many algorithms have been proposed for approximating the MAP problem [15, 5, 11, 9, 6]. Particularly, linear programming (LP) relaxation of the MAP problem has emerged as a popular technique to solve large-scale problems such as protein design and prediction problems [17, 10, 11]. Such approaches relax the constraint that the solution for the MAP problem be integral. However, for large problems such as protein design, the large size of the LP prohibits the application of standard LP-solvers [17]. To alleviate such scalability issues, convergent message passing algorithms have been introduced, which monotonically decrease the dual objective of the LP relaxation [5, 11, 9]. Convergence to the global optima is not guaranteed in general, but when the solution is integral, it can be shown to be globally optimal. The main advantage of these approaches lies in their ability to provide an upper bound on the problem and a certificate of optimality when upper bound is sufficiently close to the decoded solution.

In our work, we take a different approach to the MAP problem based on mean field methods in variational inference [16]. First, we present an alternate representation of the MAP problem by decomposing the MRF into a finite-mixture of simple Bayes nets in which maximizing the likelihood of a special variable is equivalent to solving the MAP problem. Our approach is inspired by recent developments in planning by probabilistic inference and goal-directed planning [1, 13, 14, 12]. Second, using this alternate representation, we derive the EM algorithm for approximate MAP estimation. EM increases the lower bound on the MAP assignment monotonically until convergence and lends itself naturally to a graph-based message passing implementation.

The main advantage of the EM approach lies in settings where a good approximation to MAP needs to be generated quickly. In our experiments on some of the largest protein design problems [17, 11], we show that EM increases the lower bound on MAP rapidly. This attribute of EM combined with the Max-Product LP algorithm (MPLP) [5, 11] that decreases the upper bound rapidly (as observed empirically) yields a new hybrid approach that provides quality-bounded solutions significantly faster than previous approaches. Although convergence to the global optima is not guaranteed, EM achieves an average solution quality within 95% of optimal for the protein design problems and is significantly faster than both MPLP [5, 11] and max-product (MP) [8]. We show that each iteration of EM is faster than that of max-product or MPLP by a factor related to the average degree of the graph. Empirically, the speedup factor can be as high as 30 for densely connected problems. We also show that EM is an embarrassingly parallel algorithm and can be parallelized easily to further speedup the convergence. Finally we also discuss potential pitfalls that are inherent in the EM formulation and highlight settings in which EM may not perform well.

## 2 Markov Random Fields and the MAP Problem

A pairwise Markov random field (MRF) can be described by an undirected graph $G = (V, E)$ consisting of a set of nodes, one per variable in $\boldsymbol{x} = \{x_1, \ldots, x_n\}$, and a set of edges that connect pairs of nodes. A variable can take any value from a set of possible values referred to as the domain of that variable. An edge $(i, j)$ between nodes $x_i$ and $x_j$ specifies a function $\theta_{ij}$. The joint assignment $\boldsymbol{x}$ has the probability:

$$p(\boldsymbol{x}; \boldsymbol{\theta}) = \frac{1}{Z} e^{\sum_{ij \in E} \theta_{ij}(x_i, x_j)}.$$

The MAP problem consists of finding the most probable assignment to all variables under $p(\boldsymbol{x}; \boldsymbol{\theta})$. This is equivalent to finding the complete assignment $\boldsymbol{x}$ that maximizes the function $f(\boldsymbol{x}; \boldsymbol{\theta}) = \sum_{ij \in E} \theta_{ij}(x_i, x_j)$. Before describing our formulation of the MAP problem, we first describe the marginal polytope associated with the MAP problem and its outer bound based on LP relaxation. Then we discuss the relation of our approach with these polytopes. For details, we refer to [16, 11].

Let $\boldsymbol{\mu}$ denote a vector of marginal probabilities (also called mean parameters) for each node and edge of the MRF. That is, $\boldsymbol{\mu}$ includes $\mu_i(x_i) \; \forall i \in V$ and $\mu_{ij}(x_i, x_j) \; \forall (i, j) \in E$. The set of $\boldsymbol{\mu}$ that arises from some joint distribution $p$ is referred to as the marginal polytope:

$$\mathcal{M}(G) = \{\boldsymbol{\mu} \mid \exists p(\boldsymbol{x}) \; s.t. \; p(x_i, x_j) = \mu_{ij}(x_i, x_j), \; p(x_i) = \mu_i(x_i)\} \qquad (1)$$

The MAP problem is then equivalent to solving the following LP:

$$\max_{\boldsymbol{x}} f(\boldsymbol{x}; \boldsymbol{\theta}) = \max_{\boldsymbol{\mu} \in \mathcal{M}(G)} \boldsymbol{\mu} \cdot \boldsymbol{\theta} = \max_{\boldsymbol{\mu} \in \mathcal{M}(G)} \sum_{ij \in E} \sum_{x_i x_j} \theta_{ij}(x_i, x_j) \mu_{ij}(x_i, x_j) \qquad (2)$$

It can be shown that there always exists a maximizing solution $\boldsymbol{\mu}$ which is integral and gives the optimal $\boldsymbol{x}$. Unfortunately, the number of constraints used to describe this polytope are exponential and thus it cannot be solved efficiently. To remedy this, LP relaxations are proposed that outer bound the polytope $\mathcal{M}(G)$. The relaxation weakens the global constraint that $\boldsymbol{\mu}$ arises from some common distribution $p$. Instead, only pairwise and singleton consistency is required for mean parameters as given by the following conditions:

$$\sum_{x_i} \mu_i(x_i) = 1 \; \forall i \in V \;, \sum_{\hat{x}_i} \mu_{ij}(\hat{x}_i, x_j) = \mu_j(x_j) \;, \; \sum_{\hat{x}_j} \mu_{ij}(x_i, \hat{x}_j) = \mu_i(x_i) \; \forall (i, j) \in E \qquad (3)$$

The outer bound polytope is expressed as

$$\mathcal{M}_L(G) = \{\boldsymbol{\mu} \geq 0 \mid \text{The conditions of Eq. 3 hold}\} \qquad (4)$$

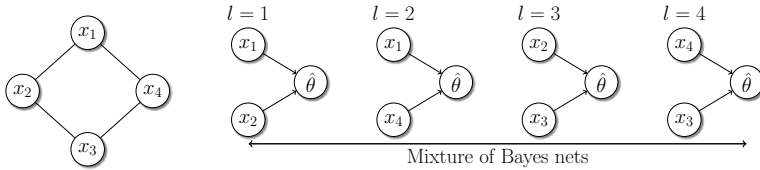

Figure 1: a) A pairwise Markov random field; b) Equivalent mixture representation

LP relaxation approaches such as MPLP [5, 11] optimize the function $\boldsymbol{\mu} \cdot \boldsymbol{\theta}$ over this outer bound $\mathcal{M}_L(G)$ and consequently yield an upper bound on the MAP. Next we describe our approach for estimating the MAP.

**Inner bound on the marginal polytope**   In the definition of marginal polytope $\mathcal{M}(G)$ (Eq. 1), no restrictions are placed on the probability distribution $p$. Consider the following class of probability distributions that factorize according to the variables of the MRF, $p'(\boldsymbol{x}) = \prod_{i=1}^{n} p_i'(x_i)$. This is similar to the mean field methods used in variational inference [16]. Our approach is to directly optimize over the following set of mean parameters $\boldsymbol{\mu}$:

$$\mathcal{M}_{lb}(G) = \{\boldsymbol{\mu} \mid \exists p'(\boldsymbol{x}) \; s.t. \; \mu_i(x_i) = p_i'(x_i) \,, \; \mu_{ij}(x_i, x_j) = p_i'(x_i)p_j'(x_j)\} \tag{5}$$

where $p'$ is the distribution that factorizes according the variables in the MRF. Clearly $\mathcal{M}_{lb}(G)$ is an inner bound over $\mathcal{M}(G)$, because in $\mathcal{M}(G)$ there is no such restriction on the class of allowed probability distributions. The optimization criterion for estimating MAP under this set is:

$$\max_{\boldsymbol{x}} f_{lb}(\boldsymbol{x}; \boldsymbol{\theta}) = \max_{\boldsymbol{\mu} \in \mathcal{M}_{lb}(G)} \boldsymbol{\mu} \cdot \boldsymbol{\theta} = \max_{\boldsymbol{\mu} \in \mathcal{M}_{lb}(G)} \sum_{ij \in E} \sum_{x_i x_j} \theta_{ij}(x_i, x_j)\mu_i(x_i)\mu_j(x_j) \tag{6}$$

Let $f_{lb}^{\star}$ denote the optimizing value for the above formulation and $f^{\star}$ for the formulation in Eq. 2. Clearly $f_{lb}^{\star} \leq f^{\star}$. A simple observation shows that indeed $f_{lb}^{\star} = f^{\star}$. The reason is that there always exists a *maximizing* $\mu \in \mathcal{M}(G)$ that is integral and thus $f^{\star}$ corresponds to an integral assignment $\boldsymbol{x}$ which is also the MAP assignment [16]. Since all the integral assignments are also allowed by the definition of the factored distribution $p'$ and $\mathcal{M}_{lb}(G)$, it follows that optimizing over $\mathcal{M}_{lb}(G)$ implies $f_{lb}^{\star} = f^{\star}$ and yields the MAP estimate. It is worth noticing that the constraints describing the set $\mathcal{M}_{lb}(G)$ are only linear in the number of nodes in the MRF and correspond to normalization constraints as opposed to the exponentially large constraint set for $\mathcal{M}(G)$.

It might appear that we have significantly reduced the space of allowed mean parameters $\boldsymbol{\mu}$ while still preserving the MAP. But the problem still remains challenging. The reduced set of parameters $\mathcal{M}_{lb}(G)$ is non-convex because of the non-linear constraint $\mu_{ij}(x_i, x_j) = \mu_i(x_i)\mu_j(x_j)$ [16]. Thus optimization over $\mathcal{M}_{lb}(G)$ cannot be done using linear programming. To alleviate this problem, we next present another reformulation of the optimization problem in Eq. 6. Then we present the Expectation Maximization (EM) algorithm [4] for this reformulation that monotonically increases the lower bound on the MAP assignment using likelihood maximization until convergence.

## 3   MAP as a Mixture of Bayes Nets

In this section we reformulate the optimization problem in Eq. 6 and recast it as the problem of likelihood maximization in a finite-mixture of simple Bayes nets. The key idea is to decompose the MRF into a mixture of simpler Bayes nets with many hidden variables – all the variables $x_i$ of the MRF. To incorporate the potential functions $\theta$'s of the MRF and achieve equivalence between the likelihood and MAP value, a special *binary reward variable* $\hat{\theta}$ is introduced with its conditional distribution proportional to potentials $\theta$. The details of the reformulation follow.

For each edge $(i, j)$ in the graph $G$ corresponding to the MRF, we create a depth-1 Bayes net. It consists of a binary reward variable $\hat{\theta}$ with its parents being the variables $x_i$ and $x_j$. The reason for calling it a reward variable will become clear later. Fig. 1(a) shows a pairwise MRF over four variables. Fig. 1(b) shows the equivalent mixture of Bayes nets for each of the four edges in this MRF. The mixture random variable $l$, which is used to identify the Bayes nets, can take values from 1 to $|E|$, the number of edges in the graph, with uniform probability. That is, if $k = |E|$, then

$P(l = i) = 1/k$ for any $1 \leq i \leq k$. In what follows, we will also use the variable $l$ to denote the corresponding edge in the MRF.

The parameters to estimate in this mixture are the marginal probabilities for each node $x_i$. That is $\boldsymbol{p} = \langle p_1, \ldots, p_n \rangle$. This step directly establishes the connection with the space of factored probability distribution $p'$ of the set $\mathcal{M}_{lb}(G)$ (see Eq. 5).

Next we set the conditional probability distribution of the variable $\hat{\theta}$ for each of the Bayes nets. This is done as follows:

$$P(\hat{\theta} = 1 | x_{l_1}, x_{l_2}, l) = \frac{\theta_l(x_{l_1}, x_{l_2}) - \theta_{min}}{\theta_{max} - \theta_{min}} \tag{7}$$

where $l$ indicates a particular Bayes net corresponding to an edge of the MRF, $x_{l_1}$ and $x_{l_2}$ are the parent variables of $\hat{\theta}$ in this Bayes net and $\theta_l$ the potential function for this edge. $\theta_{max}$ is the maximum value for any potential function $\theta$, and $\theta_{min}$ the minimum value. For example, for $l = 1$ in Fig. 1(b), $x_{l_1} = x_1$, $x_{l_2} = x_2$ and $P(\hat{\theta} = 1 | x_1, x_2, l = 1) = (\theta_{12}(x_1, x_2) - \theta_{min})/(\theta_{max} - \theta_{min})$. Note that these probabilities are nothing but the normalized potential functions $\theta_{ij}$ of the original MRF. For this reason, $\hat{\theta}$ is also called a reward variable. It is used to establish the equivalence between the MAP value and the likelihood of observing $\hat{\theta} = 1$.

The full joint for a particular Bayes net indicated by the variable $l$ is given by

$$P(\hat{\theta}, x_{l_1}, x_{l_2} | l; \boldsymbol{p}) = P(\hat{\theta} | x_{l_1}, x_{l_2}, l) p_{l_1}(x_{l_1}; \boldsymbol{p}) p_{l_2}(x_{l_2}; \boldsymbol{p}). \tag{8}$$

where $p_{l_1}$ is the marginal associated with the variable $x_{l_1}$. Let us denote the variables $(x_{l_1}, x_{l_2})$ by $x_l$ and let $\hat{\theta}_{x_l}$ denote the probability $P(\hat{\theta} = 1 | x_{l_1}, x_{l_2}, l)$, then the following theorem establishes the link between the likelihood and MAP value. $\theta_l(x_l)$ denotes the corresponding potential function $\theta_l$ of the MRF, for $l = 1$ in Fig. 1(b), $\theta_l(x_l) = \theta_{12}(x_1, x_2)$.

**Theorem 1.** *Let the CPT of binary reward variable $\hat{\theta}$ be selected such that $\hat{\theta}_{x_l} \propto \theta_l(x_l)$. Then maximizing the likelihood $L^{\boldsymbol{p}} = P(\hat{\theta} = 1; \boldsymbol{p})$ of observing the reward variable in the mixture of Bayes nets is equivalent to the MAP estimation of the original MRF.*

*Proof.* The likelihood for a single Bayes net is given by

$$L_l^{\boldsymbol{p}} = P(\hat{\theta} = 1 | l; \boldsymbol{p}) = \sum_{x_l} P(\hat{\theta} = 1, x_{l_1}, x_{l_2} | l; \boldsymbol{p}) = \sum_{x_l} \hat{\theta}_{x_l} p_{l_1}(x_{l_1}; \boldsymbol{p}) p_{l_2}(x_{l_2}; \boldsymbol{p}). \tag{9}$$

For the complete mixture, it is given by

$$L^{\boldsymbol{p}} = \sum_l P(l) L_l^{\boldsymbol{p}} = \frac{1}{k} \sum_l \sum_{x_l} \hat{\theta}_{x_l} p_{l_1}(x_{l_1}; \boldsymbol{p}) p_{l_2}(x_{l_2}; \boldsymbol{p}). \tag{10}$$

Upon substituting the definition of $\hat{\theta}_{x_l}$ from Eq. 7 and using simple algebraic manipulations, we get

$$\sum_l \sum_{x_l} \theta_l(x_l) p_{l_1}(x_{l_1}; \boldsymbol{p}) p_{l_2}(x_{l_2}; \boldsymbol{p}) = k(\theta_{min} + (\theta_{max} - \theta_{min}) L^{\boldsymbol{p}}).$$

Notice that the LHS of the above equation is the same as the optimization objective in Eq. 6. Thus we have shown that maximizing the likelihood $L^{\boldsymbol{p}}$ provides the MAP estimate. □

The above equation can also be explained intuitively in the context of goal directed planning. The RHS can be rewritten as $k(L^{\boldsymbol{p}} \theta_{max} + \bar{L}^{\boldsymbol{p}} \theta_{min})$, where $\bar{L}^{\boldsymbol{p}} = 1 - L^{\boldsymbol{p}}$. According to this formulation, there are only two rewards in the system: $\theta_{min}$ and $\theta_{max}$. The goal is to achieve the higher reward $\theta_{max}$ for each edge in the MRF. Thus maximizing the probability $L^{\boldsymbol{p}}$ of achieving this goal solves the optimization problem.

## 4   EM Algorithm for MAP Estimation

We now derive the EM algorithm [4] for maximizing the likelihood of the reward variable in the mixture of Bayes nets. In this mixture, only the reward variable is treated as observed ($\hat{\theta} = 1$); all

---

**Algorithm 1**: Graph-based message passing for MAP estimation

---

**input** : Graph $G = (V, E)$ for the MRF and potentials $\theta$ for each edge

**repeat**

    **foreach** *node $i \in V$* **do**

        **MPLP:** Send message $\gamma_{i \to j}$ to each neighbor $j \in Ne(i)$

        $\gamma_{i \to j}(x_j) \leftarrow \max_{x_i} \left[ \theta_{ij}(x_i, x_j) - \gamma_{j \to i}(x_i) + \frac{2}{|Ne(i)|+1} \sum_{k \in Ne(i)} \gamma_{k \to i}(x_i) \right]$

        Set node belief $b_i(x_i)$ to the sum of incoming messages: $b_i(x_i) = \sum_{k \in Ne(i)} \gamma_{k \to i}(x_i)$

        **EM:** Send message $\delta_{i \to j}$ to each neighbor $j \in Ne(i)$

        $\delta_{i \to j}(x_j) \leftarrow \sum_{x_i} p_i(x_i) \hat{\theta}_{x_i x_j}$

        Set marginal probability to sum of incoming messages: $p_i^\star(x_i) = p_i(x_i) \frac{\sum_{k \in Ne(i)} \delta_{k \to i}(x_i)}{C_i}$

**until** *stopping criterion is satisfied*

**MPLP**: Return complete assignment $\boldsymbol{x}$ s.t. $x_i = \text{argmax}_{\hat{x}_i} b_i(\hat{x}_i)$

**EM**    : Return complete assignment $\boldsymbol{x}$ s.t. $x_i = \text{argmax}_{\hat{x}_i} p_i(\hat{x}_i)$

---

other variables are latent. We note that EM is not guaranteed to converge to a global optimum. However, our experiments show that EM achieves an average solution quality within 95% of optimal for the standard MAP benchmark of protein design problems. We also show that the update equations for EM can be implemented efficiently using graph-based message passing and are computationally much faster than other message-passing algorithms such as max-product [8] and MPLP [5]. Below, we derive the update equations for the M-step. The E-step can be directly inferred from that. The parameters $\boldsymbol{p}$ to estimate are the marginal probabilities $p_i$ for each variable $x_i$.

**M-step:** EM maximizes the following expected complete log-likelihood for the mixture of Bayes nets. The variable $\boldsymbol{p}$ denotes the previous parameters and $\boldsymbol{p}^\star$ denotes the new parameters.

$$Q(\boldsymbol{p}, \boldsymbol{p}^\star) = \sum_l \sum_{x_l} P(\hat{\theta} = 1, x_l, l; \boldsymbol{p}) \log P(\hat{\theta} = 1, x_l, l; \boldsymbol{p}^\star) \tag{11}$$

The full joint is given by:

$$P(\hat{\theta} = 1, x_l, l; \boldsymbol{p}) = P(\hat{\theta} = 1|x_l, l) P(x_l|l; \boldsymbol{p}) P(l) = \frac{1}{k} \hat{\theta}_{x_l} p_{l_1}(x_{l_1}; \boldsymbol{p}) p_{l_2}(x_{l_2}; \boldsymbol{p})$$

We will omit the parameter $\boldsymbol{p}$ whenever the expression is unambiguous. Taking the log, we get:

$$\log P(\hat{\theta} = 1, x_l, l; \boldsymbol{p}) = \langle \text{Ind. terms of } \boldsymbol{p} \rangle + \log p_{l_1}(x_{l_1}) + \log p_{l_2}(x_{l_2}) \tag{12}$$

Substituting the above equation into the definition of $Q(\boldsymbol{p}, \boldsymbol{p}^\star)$ (Eq. 11) and discarding the terms which are independent of $\boldsymbol{p}^\star$, we get:

$$Q(\boldsymbol{p}, \boldsymbol{p}^\star) = \frac{1}{k} \sum_l \sum_{x_l} \hat{\theta}_{x_l} p_{l_1}(x_{l_1}) p_{l_2}(x_{l_2}) \{ \log p_{l_1}^\star(x_{l_1}) + \log p_{l_2}^\star(x_{l_2}) \} \tag{13}$$

Upon simplifying the above equation by grouping together the terms associated with the variables $x_i$ of the MRF, we get:

$$Q(\boldsymbol{p}, \boldsymbol{p}^\star) = \frac{1}{k} \sum_{i=1}^n \sum_{x_i} p_i(x_i) \log p_i^\star(x_i) \sum_{j \in Ne(i)} \sum_{x_j} \hat{\theta}_{x_i x_j} p_j(x_j) \tag{14}$$

where $Ne(i)$ denotes the set of immediate neighbors of the node $i$ in the MRF graph. The above expression can be easily maximized by maximizing for variables $x_i$'s individually. The final update equation for the marginals is given by:

$$p_i^\star(x_i) = \frac{p_i(x_i) \sum_{j \in Ne(i)} \sum_{x_j} \hat{\theta}_{x_i x_j} p_j(x_j)}{C_i} \tag{15}$$

where $C_i$ is the normalization constant for variable $x_i$, and $\hat{\theta}_{x_i x_j}$ is the normalized reward:

$$P(\hat{\theta} = 1|x_i, x_j, l) = (\theta_{ij}(x_i, x_j) - \theta_{min})/(\theta_{max} - \theta_{min}).$$

Algorithm 1 shows the graph-based message passing technique for both EM and MPLP. For both EM and MPLP, parameters are initialized randomly. The rest of the steps are self-explanatory.

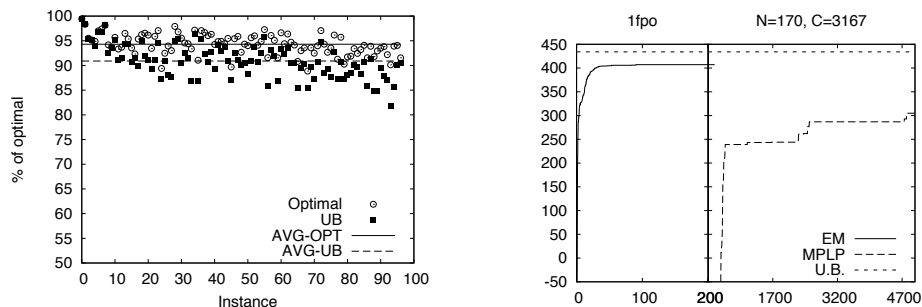

Figure 2: a) Quality achieved by EM for all protein design instances. b) Quality for the largest instance '1fpo'; $x$-axis denotes time (in sec.), $y$-axis denotes the quality achieved.

**Complexity analysis and implementation** Consider a single message $\gamma$ sent out by a node in MPLP. The complexity of computing $\gamma$ is $O(d^2 \cdot deg)$, where $d$ is the domain size of the variables, and $deg$ represents the average degree of the graph or the average number of neighbors of a node. For EM, this complexity is only $O(d^2)$. Therefore, the computational complexity of each iteration of EM is lower than that of MPLP by a factor of $deg$. The same result holds for max-product, because its message-passing structure is similar to that of MPLP. The average number of neighbors in dense graphs such as the ones encountered in protein-design problems can be as high as 30. This makes EM significantly faster than the previous approaches as we demonstrate empirically in the next section.

EM's simple message passing scheme also facilitates a very efficient parallel implementation. In particular, all the $\delta$ messages for the current iteration in Alg. 1 can be computed in *parallel* for each node $i$, because they depend only on the parameters from the previous iteration. In contrast, MPLP follows a block coordinate descent strategy in which optimization is performed over a subset of variables, keeping all the other variables fixed [5]. Therefore, opportunities for parallelism in the current implementations of MPLP are limited.

## 5 Experiments

Our first set of experiments are on the protein design problems (total of 97 instances), which are described in [17]. In these problems, given a desired backbone structure of the protein, the task is to find a sequence of amino-acids that is as stable as possible or has the lowest energy. This problem can be represented as finding the MAP configuration in an MRF. These problems are particularly hard and dense with up to 170 variables, each having a large domain size of up to 150 values. We compare performance with the MPLP algorithm as described in [5, 11] and with max-product [8]. We used the standard setting for MPLP – first it is run with edge based clusters for 1000 iterations [5] and then clusters of size 3 are added to tighten the LP relaxation [11]. EM was implemented in JAVA. To speedup the convergence of EM, we used a simple modification of the M-step as described in [12].

All our experiments were done on a Mac Pro with dual quad-core processor and 4GB RAM. All algorithms used only a single processor for computation. We note that another clustering-based improvement of MPLP is presented in [9]. Such clusters can be similarly incorporated into the EM algorithm, which currently does not use any clusters. Therefore, comparisons with such clustering techniques are left for future work.

The main purpose of our experiments is to show that EM achieves high solution quality, much more quickly than MPLP or max-product. Therefore EM provides a good alternative, particularly when fast near-optimal solutions are desired. As reported in [11], for protein design problems solved exactly, mean running time was 9.7 hours. For all the problems, instead of running MPLP until the near-optimal solution is found, we used a fixed cutoff of 5000 sec. For all the problems, we ran EM and max-product for 1500 iterations. For EM, different runs were initialized randomly and the best of 5-runs is plotted. Empirically, EM achieves a solution quality within 95% of optimal on average much faster than MPLP. The longest time EM took for any protein design instance was 352 sec. for the '1fpo' instance (Fig. 2(b)).

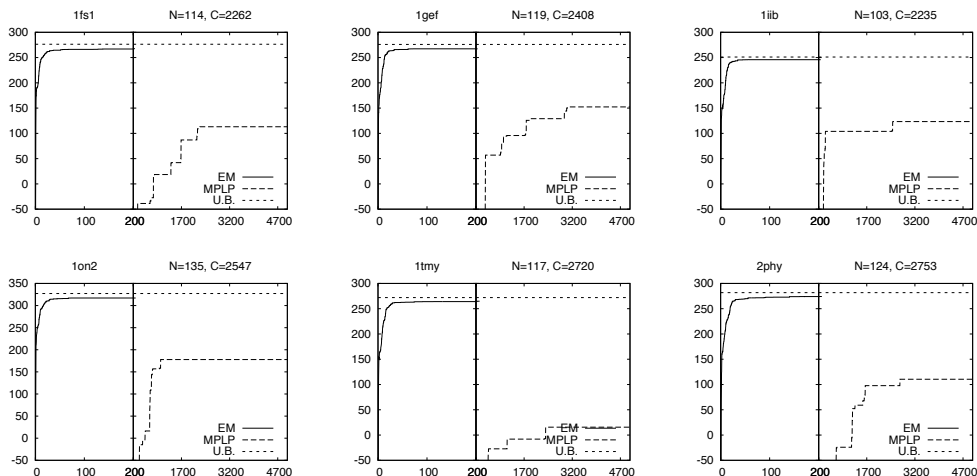

Figure 3: Quality comparison with MPLP for six of the largest protein design instances. The $x$-axis denotes time (in sec.) and the $y$-axis denotes the quality achieved.

Fig. 2(a) shows the solution quality EM achieves for all the instances in 1500 iterations. Since a tight upper bound for all the problems except the instance '1fpo' is known [11], we show the percentage of optimal EM achieves. The legend titled 'Optimal' in Fig. 2(a) shows this value. For the unsolved instance '1fpo', we use the best known upper bound MPLP achieved in 10 hours ($\approx 434$). As it is clear from this graph, EM achieves near-optimal solution quality for all the instances, within 95% on average. To show empirically that MPLP decreases the upper bound quickly, we also show the percentage of solution quality EM achieves when instead of using the best known upper bound, we use the upper bound provided by MPLP after 1,000 iterations. The legend in Fig. 2(a) titled 'UB' shows this percentage. Even using this bound, EM achieves a quality within 91% on average (legend 'AVG-UB'). This further suggests that combining EM's ability to rapidly increase the lower bound and MPLP's ability to decrease the upper bound quickly, is a good way to create a hybrid approach that can provide provably near-optimal solutions much faster.

Fig. 2(b) shows the quality achieved by EM and MPLP as a function of time for the largest instance '1fpo'. To show the convergence curve of EM clearly, the plot uses a different scale for time $T \leq 200$ and the rest. This graph also shows that EM provides a much better solution quality, much faster than the MPLP. Legend 'U.B' denotes the best known upper bound. Empirically, we noticed that the main advantage of the EM approach was for problems which were large in size having many variables. For smaller problems, EM and MPLP were comparable in performance. Fig. 3 shows the quality comparisons with time for some of the largest protein design instances. Each graph title shows the instance name, $N$ denotes the number of variables in the MRF, $C$ denotes the number of potential functions $\theta$ or edges in the graph. For all these problems, EM provided near-optimal solution quality and is significantly faster than MPLP.

We also compared EM with max-product. Table 1 shows this comparison for some of the largest protein design instances. In this table, MP Quality denotes the best quality max-product achieved in 1500 iterations, Time/Iteration denotes the time and iteration number when it was achieved for the first time. Again, EM outperforms max-product by a significant margin, achieving a higher solution quality much faster. For some of the problems, such as '1tmy' and '1or7', the quality achieved by EM was much higher. Also, for none of these problems max-product converged. This may be due to the fact that these are highly constrained problems with many cycles in the graph. The average degree of a node for these problems is very high, e.g., $\approx 37$ for '1fpo'. The time required per iteration of max-product was 11 sec. for this instance. Therefore the predicted time per EM's iteration is $11/37 \approx 0.298$ sec.; the actual time for EM was $0.235$ sec. The same result holds for other instances as well. This is consistent with the complexity analysis in Sec. 4.

We also tested EM on the protein prediction problems [17, 11] which are simpler and sparser than the protein design problems. The LP relaxations in this case can be solved even by the standard LP solvers unlike the protein design problems [17]. Surprisingly, EM does not work well on these

| Instance | MP Quality | Time/Iteration | EM Quality | Time/Iteration | U.B. |
|---|---|---|---|---|---|
| 1fs1 | 268.3 | 3628.5/344 | 267.6 | 202.3/1220 | 276.4 |
| 1gef | 239.8 | 13938.9/1331 | 267.3 | 71.6/428 | 276.1 |
| 1bkb | 272.4 | 10928.9/965 | 288.2 | 250.1/1462 | 292.8 |
| 1iib | 236.2 | 11493.1/1099 | 245.7 | 78.1/442 | 251.1 |
| 1on2 | 314.7 | 11628.34/1226 | 317.1 | 146.6/807 | 327.23 |
| 1tmy | 202.9 | 99.5/8 | 264.8 | 222.4/1067 | 272.1 |
| 1or7 | 368.1 | 234.9/22 | 410.2 | 240.8/1087 | 419.3 |
| 1fpo | 406.2 | 9072.7/791 | 407.1 | 263.6/1125 | 434 |

Table 1: Solution quality and time (in sec.) comparison between EM and Max-Product (MP). U.B. denotes the best known upper bound.

problems. For the hardest instance '1a8i' (812 variables, 10124 edges, edge density=.03), MPLP achieves the near optimal value of 73, whereas EM could only achieve a value of $-374$. The reason for this lies in the reward structure of the problem or the values $\theta_{min}$ and $\theta_{max}$. For this problem $\theta_{min} = -5770.96$ and $\theta_{max} = 3.88$. As shown earlier, EM works with the normalized rewards assigning 0 to the minimum reward $-5770.96$, and 1 to the maximum reward $3.88$. This dramatic scaling of the reward is particularly problematic for EM as shown below.

According to Thm. 1, the log-likelihood EM converges to is $-2.949 \times 10^{-4}$. For EM to achieve the value 73, the log-likelihood should be $-2.913 \times 10^{-4}$. However the drastic scaling of the reward causes this minor difference to significantly affect solution quality. In such settings, EM may not work well. In contrast, for the largest protein design instance '1fpo', the minimum reward is $-59.2$ and maximum is $4.37$. We also experimented on a $10 \times 10$ grid graph with 5 values per variable using the Potts model similar to [5]. We randomly generated 100 instances and found that EM achieved good solution quality, within 95% of optimal on average. The difference between the maximum and minimum reward in these problems was less than 5, with a typical setting: $\theta_{min} \approx -2.5$, $\theta_{max} \approx 2.5$. This further supports the previous analysis.

# 6 Conclusion

A number of techniques have been developed recently to find the MAP assignment of Markov random fields. Particularly successful are approaches based on LP relaxation of the MAP problem such as MPLP. Such approaches minimize an upper bound relatively quickly, but take much longer to find a good solution. In contrast, our proposed formulation seeks to provide good quality solutions quickly by directly maximizing a lower bound on the MAP value over the inner bound on the marginal polytope. The proposed Expectation Maximization (EM) algorithm increases this lower bound monotonically by likelihood maximization and is guaranteed to converge. Furthermore, EM's update equations can be efficiently implemented using a graph-based message passing paradigm.

Although EM may get stuck at a local optimum, our empirical results on the protein design dataset show that EM performs very well, producing solutions within 95% of optimal on average. EM achieves such high solution quality significantly faster than MPLP or max-product for many large protein design problems. Another significant advantage EM enjoys is the ease of parallelization. Using advanced parallel computing paradigms such as Google's MapReduce [2] can further speedup the algorithm with little additional effort. Finally, we examined a setting in which EM may not work well due to a large gap between the minimum and maximum reward. Our ongoing efforts include incorporating some of the advanced clustering techniques based on LP relaxation of the MAP problem with the EM method, and designing heuristics that can help EM avoid getting stuck in local optima for problems with large variations in the reward structure.

# 7 Acknowledgment

We thank anonymous reviewers for their helpful suggestions. Support for this work was provided in part by the National Science Foundation Grant IIS-0812149 and by the Air Force Office of Scientific Research Grant FA9550-08-1-0181.

# References

[1] H. Attias. Planning by probabilistic inference. In *Proc. of the 9th Int. Workshop on Artificial Intelligence and Statistics*, 2003.

[2] J. Dean and S. Ghemawat. MapReduce: a flexible data processing tool. *Communications of the ACM*, 53(1):72–77, 2010.

[3] R. Dechter. *Constraint Processing*. Morgan Kaufmann Publishers Inc., San Francisco, CA, USA, 2003.

[4] A. P. Dempster, N. M. Laird, and D. B. Rubin. Maximum likelihood from incomplete data via the EM algorithm. *Journal of the Royal Statistical society, Series B*, 39(1):1–38, 1977.

[5] A. Globerson and T. Jaakkola. Fixing Max-Product: Convergent message passing algorithms for MAP LP-relaxations. In *Advances in Neural Information Processing Systems*, 2007.

[6] K. Jung, P. Kohli, and D. Shah. Local rules for global MAP: When do they work? In *Advances in Neural Information Processing Systems*, 2009.

[7] C. D. Manning and H. Schütze. *Foundations of statistical natural language processing*. MIT Press, Cambridge, MA, USA, 1999.

[8] J. Pearl. *Probabilistic Reasoning in Intelligent Systems*. Morgan Kaufmann Publishers Inc., 1988.

[9] D. Sontag, A. Globerson, and T. Jaakkola. Clusters and coarse partitions in LP relaxations. In *Advances in Neural Information Processing Systems*, pages 1537–1544, 2008.

[10] D. Sontag and T. Jaakkola. New outer bounds on the marginal polytope. In *Advances in Neural Information Processing Systems*, 2007.

[11] D. Sontag, T. Meltzer, A. Globerson, T. Jaakkola, and Y. Weiss. Tightening LP relaxations for MAP using message passing. In *Proc. of Uncertainty in Artificial Intelligence*, pages 503–510, 2008.

[12] M. Toussaint, L. Charlin, and P. Poupart. Hierarchical POMDP controller optimization by likelihood maximization. In *Proc. of Uncertainty in Artificial Intelligence*, pages 562–570, 2008.

[13] M. Toussaint, S. Harmeling, and A. Storkey. Probabilistic inference for solving (PO)MDPs. Technical Report EDIINF-RR-0934, University of Edinburgh, School of Informatics, 2006.

[14] M. Toussaint and A. J. Storkey. Probabilistic inference for solving discrete and continuous state markov decision processes. In *Proc. of the International Conference on Machine Learning*, pages 945–952, 2006.

[15] M. Wainwright, T. Jaakkola, and A. Willsky. MAP estimation via agreement on (hyper)trees: Message-passing and linear programming approaches. *IEEE Transactions on Information Theory*, 51:3697–3717, 2002.

[16] M. J. Wainwright and M. I. Jordan. Graphical models, exponential families, and variational inference. *Foundations and Trends in Machine Learning*, 1(1-2):1–305, 2008.

[17] C. Yanover, T. Meltzer, Y. Weiss, P. Bennett, and E. Parrado-hernndez. Linear programming relaxations and belief propagation – an empirical study. *Journal of Machine Learning Research*, 7:2006, 2006.

